# Fundamental Limitations of Spectral Clustering

**Boaz Nadler**,* **Meirav Galun**
Department of Applied Mathematics and Computer Science
Weizmann Institute of Science, Rehovot, Israel 76100
`boaz.nadler,meirav.galun@weizmann.ac.il`

## Abstract

Spectral clustering methods are common graph-based approaches to clustering of data. Spectral clustering algorithms typically start from *local* information encoded in a weighted graph on the data and cluster according to the *global* eigenvectors of the corresponding (normalized) similarity matrix. One contribution of this paper is to present fundamental limitations of this general local to global approach. We show that based only on local information, the normalized cut functional is not a suitable measure for the quality of clustering. Further, even with a suitable similarity measure, we show that the first few eigenvectors of such adjacency matrices cannot successfully cluster datasets that contain structures at different scales of size and density. Based on these findings, a second contribution of this paper is a novel diffusion based measure to evaluate the coherence of individual clusters. Our measure can be used in conjunction with any bottom-up graph-based clustering method, it is scale-free and can determine coherent clusters at all scales. We present both synthetic examples and real image segmentation problems where various spectral clustering algorithms fail. In contrast, using this coherence measure finds the expected clusters at all scales.

**Keywords:** Clustering, kernels, learning theory.

## 1   Introduction

Spectral clustering methods are common graph-based approaches to (unsupervised) clustering of data. Given a dataset of $n$ points $\{\boldsymbol{x}_i\}_{i=1}^n \subset \mathbb{R}^p$, these methods first construct a weighted graph $G = (V, W)$, where the $n$ points are the set of nodes $V$ and the weighted edges $W_{i,j}$ are computed by some local symmetric and non-negative similarity measure. A common choice is a Gaussian kernel with width $\sigma$, where $\| \cdot \|$ denotes the standard Euclidean metric in $\mathbb{R}^p$

$$W_{i,j} = \exp\left(-\frac{\|\boldsymbol{x}_i - \boldsymbol{x}_j\|^2}{2\sigma^2}\right) \tag{1}$$

In this framework, clustering is translated into a graph partitioning problem. Two main spectral approaches for graph partitioning have been suggested. The first is to construct a normalized cut (conductance) functional to measure the quality of a partition of the graph nodes $V$ into $k$ clusters[1, 2]. Specifically, for a 2-cluster partition $V = S \cup (V \setminus S)$ minimizing the following functional is suggested in [1]

$$\phi(S) = \sum_{i \in S, j \in V \setminus S} W_{i,j} \left[ \frac{1}{a(S)} + \frac{1}{a(V \setminus S)} \right] \tag{2}$$

where $a(S) = \sum_{i \in S, j \in V} W_{i,j}$. While extensions of this functional to more than two clusters are possible, both works suggest a recursive top-down approach where additional clusters are found by

minimizing the same clustering functional on each of the two subgraphs. In [3] the authors also propose to augment this top-down approach by a bottom-up aggregation of the sub-clusters.

As shown in [1] minimization of (2) is equivalent to $\max_{\boldsymbol{y}}(\boldsymbol{y}^T W \boldsymbol{y})/(\boldsymbol{y}^T D \boldsymbol{y})$, where $D$ is a diagonal $n \times n$ matrix with $D_{i,i} = \sum_j W_{i,j}$, and $\boldsymbol{y}$ is a vector of length $n$ that satisfies the constraints $\boldsymbol{y}^T D \mathbf{1} = 0$ and $\boldsymbol{y}_i \in \{1, -b\}$ with $b$ some constant in $(0, 1)$. Since this maximization problem is NP-hard, both works relax it by allowing the vector $\boldsymbol{y}$ to take on real values. This approximation leads to clustering according to the eigenvector with second largest eigenvalue of the normalized graph Laplacian, $W\boldsymbol{y} = \lambda D \boldsymbol{y}$. We note that there are also graph partitioning algorithms based on a non-normalized functional leading to clustering according to the second eigenvector of the standard graph Laplacian matrix $D - W$, also known as the Fiedler vector [4].

A second class of spectral clustering algorithms does not recursively employ a single eigenvector, but rather proposes to map the original data into the first $k$ eigenvectors of the normalized adjacency matrix (or a matrix similar to it) and then apply a standard clustering algorithm such as $k$-means on these new coordinates, see for example [5]-[11] and references therein. In recent years, much theoretical work was done to justify this approach. Belkin and Niyogi [8] showed that for data uniformly sampled from a manifold, these eigenvectors approximate the eigenfunctions of the Laplace Beltrami operator, which give an optimal low dimensional embedding under a certain criterion. Optimality of these eigenvectors, including rotations, was derived in [9] for multiclass spectral clustering. Probabilistic interpretations, based on the fact that these eigenvectors correspond to a random walk on the graph were also given by several authors [11]-[15]. Limitations of spectral clustering in the presence of background noise and multiscale data were noted in [10, 16], with suggestions to replace the uniform $\sigma^2$ in eq. (1) with a location dependent scale $\sigma(\boldsymbol{x}_i)\sigma(\boldsymbol{x}_j)$.

The aim of this paper is to present *fundamental limitations of spectral clustering methods*, and propose a novel *diffusion based coherence measure* to evaluate the internal consistency of individual clusters. First, in Section 2 we show that based on the isotropic local similarity measure (1), the NP-hard normalized cut criterion may not be a suitable global functional for data clustering. We construct a simple example with only two clusters, where we prove that the minimum of this functional does not correspond to the natural expected partitioning of the data into its two clusters. Further, in Section 3 we show that spectral clustering suffers from additional limitations, even with a suitable similarity measure. Our theoretical analysis is based on the probabilistic interpretation of spectral clustering as a random walk on the graph and on the intimate connection between the corresponding eigenvalues and eigenvectors and the characteristic relaxation times and processes of this random walk. We show that similar to Fourier analysis, spectral clustering methods are global in nature. Therefore, even with a location dependent $\sigma(\boldsymbol{x})$ as in [10], these methods typically fail to simultaneously identify clusters at different scales. Based on this analysis, we present in Section 4 simple examples where spectral clustering fails. We conclude with Section 5, where we propose a novel diffusion based coherence measure. This quantity measures the coherence of a set of points as all belonging to a single cluster, by comparing the relaxation times on the set and on its suggested partition. Its main use is as a decision tool whether to divide a set of points into two subsets or leave it intact as a single coherent cluster. As such, it can be used in conjunction with either top-down or bottom-up clustering approaches and may overcome some of their limitations. We show how use of this measure correctly clusters the examples of Section 4, where spectral clustering fails.

## 2 Unsuitability of normalized cut functional with local information

As reported in the literature, clustering by approximate minimization of the functional (2) performs well in many cases. However, a theoretical question still remains: Under what circumstances is this functional indeed a good measure for the quality of clustering ? Recall that the basic goal of clustering is to group together highly similar points while setting apart dissimilar ones. Yet this similarity measure is typically based only on *local* information as in (1). Therefore, the question can be rephrased - is local information sufficient for global clustering ?

While this *local to global* concept is indeed appealing, we show that it does not work in general. We construct a simple example where local information is insufficient for correct clustering according to the functional (2). Consider data sampled from a mixture of two densities in two dimensions

$$p(\boldsymbol{x}) = p(x_1, x_2) = \frac{1}{2} \left[ p_{L,\varepsilon}(x_1, x_2) + p_G(x_1, x_2) \right] \tag{3}$$

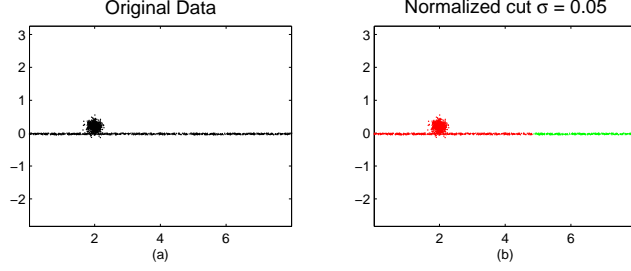

Figure 1: A dataset with two clusters and result of normalized cut algorithm [2]. Other spectral clustering algorithms give similar results.

where $p_{L,\varepsilon}$ denotes uniform density in a rectangular region $\Omega = \{(x_1, x_2) \,|\, 0 < x_1 < L, -\varepsilon < x_2 < 0\}$ of length $L$ and width $\varepsilon$, and $p_G$ denotes a Gaussian density centered at $(\mu_1, \mu_2)$ with diagonal covariance matrix $\rho^2 I$. In fig. 1(a) a plot of $n = 1400$ points from this density is shown with $L = 8, \varepsilon = 0.05 \ll L, (\mu_1, \mu_2) = (2, 0.2)$ and $\rho = 0.1$. Clearly, the two clusters are the Gaussian ball and the rectangular strip $\Omega$.

However, as shown in fig. 1(b), clustering based on the second eigenvector of the normalized graph Laplacian with weights $W_{i,j}$ given by (1) partitions the points somewhere along the long strip instead of between the strip and the Gaussian ball. We now show that this result is not due to the approximation of the NP-hard problem but rather a feature of the original functional (2). Intuitively, the failure of the normalized cut criterion is clear. Since the overlap between the Gaussian ball and the rectangular strip is larger than the width of the strip, a cut that separates them has a higher penalty than a cut somewhere along the thin strip.

To show this mathematically, we consider the penalty of the cut due to the numerator in (2) in the limit of a large number of points $n \to \infty$. In this population setting, as $n \to \infty$ each point has an infinite number of neighbors, so we can consider the limit $\sigma \to 0$. Upon normalizing the similarity measure (1) by $1/2\pi\sigma^2$, the numerator is given by

$$Cut(\Omega_1) = \lim_{n \to \infty} \frac{1}{|V|} \sum_{\boldsymbol{x} \in \Omega_1} \sum_{\boldsymbol{y} \in \Omega_2} W_{i,j} = \frac{1}{2\pi\sigma^2} \int_{\Omega_1} \int_{\Omega_2} p(\boldsymbol{x})p(\boldsymbol{y}) e^{-\|\boldsymbol{x}-\boldsymbol{y}\|^2/2\sigma^2} d\boldsymbol{x} d\boldsymbol{y} \quad (4)$$

where $\Omega_1, \Omega_2 \subset \mathbb{R}^2$ are the regions of the two clusters. For $\varepsilon \ll L$, a vertical cut of the strip at location $x = x_1$ far away from the ball ($|\boldsymbol{x}_1 - \boldsymbol{x}_0| \gg \rho$) gives

$$Cut(x > x_1) \simeq \lim_{\sigma \to 0} \int_0^\infty \int_{-\infty}^0 \frac{1}{L^2} \frac{1}{2\pi\sigma^2} e^{-(x-x')^2/2\sigma^2} dx dx' = \frac{1}{2\pi L^2} \quad (5)$$

A similar calculation shows that for a horizontal cut at $y = 0$,

$$Cut(y > 0) \simeq \frac{1}{L} \frac{e^{-\mu_2^2/2\rho^2}}{\sqrt{8\pi}\rho} \quad (6)$$

Finally, note that for a vertical cut far from the rectangle boundary $\partial\Omega$, the denominators of the two cuts in eq. (2) have the same order of magnitude. Therefore, if $L \gg \rho$ and $\mu_2/\rho = O(1)$ the horizontal cut between the ball and the strip has *larger* normalized penalty than a vertical cut of the strip. This analysis explains the numerical results in fig. 1(b). Other spectral clustering algorithms that use two eigenvectors, including those that take a local scale into account, also fail to separate the ball from the strip and yield similar results to fig.1(b). A possible solution to this problem is to introduce multiscale anisotropic features that capture the geometry and dimensionality of the data in the similarity metric. In the context of image and texture segmentation, the need for multiscale features is well known [17, 18, 19]. Our example highlights its importance in general data clustering.

## 3   Additional Limitations of Spectral Clustering Methods

An additional problem with recursive bi-partitioning is the need of a saliency criterion when required to return $k > 2$ clusters. Consider, for example a dataset which contains $k = 3$ clusters. After

the first cut, the recursive algorithm should decide which subgraph to further partition and which to leave intact. A common approach that avoids this decision problem is to directly find three clusters by using the first three eigenvectors of $W\boldsymbol{v} = \lambda D\boldsymbol{v}$. Specifically, denote by $\{\lambda_j, \boldsymbol{v}_j\}$ the set of eigenvectors of $W\boldsymbol{v} = \lambda D\boldsymbol{v}$ with eigenvalues sorted in decreasing order, and denote by $\boldsymbol{v}_j(\boldsymbol{x}_i)$ the $i$-th entry (corresponding to the point $\boldsymbol{x}_i$) in the $j$-th eigenvector $\boldsymbol{v}_j$. Many algorithms propose to map each point $\boldsymbol{x}_i \in \mathbb{R}^p$ into $\Psi(\boldsymbol{x}_i) = (\boldsymbol{v}_1(\boldsymbol{x}_i), \ldots, \boldsymbol{v}_k(\boldsymbol{x}_i)) \in \mathbb{R}^k$, and apply simple clustering algorithms to the points $\Psi(\boldsymbol{x}_i)$ [8, 9, 12]. Some works [6, 10] use the eigenvectors $\tilde{\boldsymbol{v}}_j$ of $D^{-1/2}WD^{-1/2}$ instead, related to the ones above via $\tilde{\boldsymbol{v}}_j = D^{1/2}\boldsymbol{v}_j$.

We now show that spectral clustering that uses the first $k$ eigenvectors for finding $k$ clusters also suffers from fundamental limitations. Our starting point is the observation that $\boldsymbol{v}_j$ are also eigenvectors of the Markov matrix $M = D^{-1}W$ [13, 12]. Assuming the graph is connected, the largest eigenvalue is $\lambda_1 = 1$ with $|\lambda_j| < 1$ for $j > 1$. Therefore, regardless of the initial condition the random walk converges to the unique equilibrium distribution $\pi_s$, given by $\pi_s(i) = D_{i,i}/\sum_j D_{j,j}$. Moreover, as shown in [13], the Euclidean distance between points mapped to these eigenvectors is equal to a so called 'diffusion distance' between points on the graph,

$$\sum_j \lambda_j^{2t}\left(\boldsymbol{v}_j(\boldsymbol{x}) - \boldsymbol{v}_j(\boldsymbol{y})\right)^2 = \|p(\boldsymbol{z}, t \,|\, \boldsymbol{x}) - p(\boldsymbol{z}, t \,|\, \boldsymbol{y})\|_{L_2(1/\pi_s)}^2 \tag{7}$$

where $p(\boldsymbol{z}, t \,|\, \boldsymbol{x})$ is the probability distribution of a random walk at time $t$ given that it started at $\boldsymbol{x}$, $\pi_s$ is the equilibrium distribution, and $\|\cdot\|_{L_2(w)}$ is the weighted $L_2$ norm with weight $w(\boldsymbol{z})$. Therefore, the eigenvalues and eigenvectors $\{\lambda_j, \boldsymbol{v}_j\}$ for $j > 1$, capture the characteristic *relaxation times* and processes of the random walk on the graph towards equilibrium. Since most methods use the first few eigenvector coordinates for clustering, it is instructive to study the properties of these relaxation times and of the corresponding eigenvectors.

We perform this analysis under the following statistical model: we assume that the points $\{\boldsymbol{x}_i\}$ are random samples from a smooth density $p(\boldsymbol{x})$ in a smooth domain $\Omega \subset \mathbb{R}^p$. We write the density in Boltzmann form $p(\boldsymbol{x}) = e^{-U(\boldsymbol{x})/2}$ and denote $U(\boldsymbol{x})$ as the potential. As described in [13], in the limit $n \to \infty$, $\sigma \to 0$, the random walk with transition matrix $M$ on the graph of points sampled from this density converges to a stochastic differential equation (SDE)

$$\dot{\boldsymbol{x}}(t) = -\nabla U(\boldsymbol{x}) + \sqrt{2}\dot{\boldsymbol{w}}(t) \tag{8}$$

where $\boldsymbol{w}(t)$ is standard white noise (Brownian motion), and the right eigenvectors of the matrix $M$ converge to the eigenfunctions of the following Fokker-Planck operator

$$\mathcal{L}\psi(\boldsymbol{x}) \equiv \Delta\psi - \nabla\psi \cdot \nabla U = -\mu\psi(\boldsymbol{x}) \tag{9}$$

defined for $\boldsymbol{x} \in \Omega$ with reflecting boundary conditions on $\partial\Omega$. This operator is non-positive and its eigenvalues are $\mu_1 = 0 < \mu_2 \le \mu_3 \le \ldots$. The eigenvalues $-\mu_j$ of $\mathcal{L}$ and the eigenvalues $\lambda_j$ of $M$ are related by $\mu_j = \lim_{n \to \infty, \sigma \to 0}(1 - \lambda_j)/\sigma$. Therefore the top eigenvalues of $M$ correspond to the smallest of $\mathcal{L}$. Eq. (7) shows that these eigenfunctions and eigenvalues capture the leading characteristic relaxation processes and time scales of the SDE (8). These have been studied extensively in the literature [20], and can give insight into the success and limitations of spectral clustering [13]. For example, if $\Omega = \mathbb{R}^p$ and the density $p(\boldsymbol{x})$ consists of $k$ highly separated Gaussian clusters of roughly equal size ($k$ clusters), then there are exactly $k$ eigenvalues very close or equal to zero, and their corresponding eigenfunctions are approximately piecewise constant in each of these clusters. Therefore, in this setting spectral clustering with $k$ eigenvectors works very well.

To understand the limitations of spectral clustering, we now explicitly analyze situations with clusters at different scales of size and density. For example, consider a density with three isotropic Gaussian clusters: one large cloud (cluster #1) and two smaller clouds (clusters 2 and 3). These correspond to one wide well and two narrow wells in the potential $U(\boldsymbol{x})$. A representative 2-D dataset drawn from such a density is shown in fig. 2 (top left).

The SDE (8) with this potential has a few characteristic time scales which determine the structure of its leading eigenfunctions. The slowest one is the mean passage time between cluster 1 and clusters 2 or 3, approximately given by [20]

$$\tau_{1,2} = \frac{2\pi}{\sqrt{|U''_{min}U''_{max}|}}\, e^{(U(x_{max}) - U(x_{min}))} \tag{10}$$

where $\boldsymbol{x}_{min}$ is the bottom of the deepest well, $\boldsymbol{x}_{max}$ is the saddle point of $U(\boldsymbol{x})$, and $U''_{min}, U''_{max}$ are the second derivatives at these points. Eq. (10), also known as Arrhenius or Kramers formula of chemical reaction theory, shows that the mean first passage time is exponential in the barrier height [20]. The corresponding eigenfunction $\psi_2$ is approximately piecewise constant inside the large well and inside the two smaller wells with a sharp transition near the saddle point $\boldsymbol{x}_{max}$. This eigenfunction easily separates cluster 1 from clusters 2 and 3 (see top center panel in fig. 2).

A second characteristic time is $\tau_{2,3}$, the mean first passage time between clusters 2 and 3, also given by a formula similar to (10). If the potential barrier between these two wells is much smaller than between wells 1 and 2, then $\tau_{2,3} \ll \tau_{1,2}$. A third characteristic time is the equilibration time inside cluster 1. To compute it we consider a diffusion process only inside cluster 1, e.g. with an isotropic parabolic potential of the form $U(\boldsymbol{x}) = U(\boldsymbol{x}_1) + U''_1 \|\boldsymbol{x} - \boldsymbol{x}_1\|^2 / 2$, where $\boldsymbol{x}_1$ is the bottom of the well. In 1-D the eigenvalues and eigenfunctions are given by $\mu_k = (k-1)U''_1$, with $\psi_k(\boldsymbol{x})$ a polynomial of degree $k-1$. The corresponding intra-well relaxation times are given by $\tau_k^R = 1/\mu_{k+1}$ ($k \geq 1$).

The key point in our analysis is that if the equilibration time inside the wide well is *slower* than the mean first passage time between the two smaller wells, $\tau_1^R > \tau_{2,3}$, then the third eigenfunction of $\mathcal{L}$ captures the relaxation process inside the large well and is approximately constant inside the two smaller wells. This eigenfunction cannot separate between clusters 2 and 3. Moreover, if $\tau_2^R = \tau_1^R/2$ is still larger than $\tau_{2,3}$ then even the next leading eigenfunction captures the equilibration process inside the wide well, see a plot of $\psi_3, \psi_4$ in fig. 2 (rows 1,2). Therefore, even this next eigenfunction is not useful for separating the two small clusters. In the example of fig. 2, only $\psi_5$ separates these two clusters.

This analysis shows that when confronted with clusters of different scales, corresponding to a multiscale landscape potential, standard spectral clustering which uses the first $k$ eigenvectors to find $k$ clusters will fail. We present explicit examples in Section 4 below. The fact that spectral clustering with a single scale $\sigma$ may fail to correctly cluster multiscale data was already noted in [10, 16]. To overcome this failure, [10] proposed replacing the uniform $\sigma^2$ in eq. (1) with $\sigma(\boldsymbol{x}_i)\sigma(\boldsymbol{x}_j)$ where $\sigma(\boldsymbol{x})$ is proportional to the local density at $\boldsymbol{x}$. Our analysis can also provide a probabilistic interpretation to their method. In a nutshell, the effect of this scaling is to *speed up* the diffusion process at regions of low density, thus changing some of its characteristic times. If the larger cluster has low density, as in the examples in their paper, this approach is successful as it decreases $\tau_1^R$. However, if the large cluster has a high density (comparable to the density of the small clusters), this approach is not able to overcome the limitations of spectral clustering, see fig. 3. Moreover, this approach may also fail in the case of uniform density clusters defined solely by geometry (see fig. 4).

## 4 Examples

We illustrate the theoretical analysis of Section 3 with three examples, all in 2-D. In the first two examples, the $n$ points $\{\boldsymbol{x}_i\} \subset \mathbb{R}^2$ are random samples from the following mixture of three Gaussians

$$\alpha_1 N(\boldsymbol{x}_1, \sigma_1^2 I) + \alpha_2 N(\boldsymbol{x}_2, \sigma_2^2 I) + \alpha_3 N(\boldsymbol{x}_3, \sigma_3^2 I) \tag{11}$$

with centers $\boldsymbol{x}_i$ isotropic standard deviations $\sigma_i$ and weights $\alpha_i$ ($\sum_i \alpha_i = 1$). Specifically, we consider one large cluster with $\sigma_1 = 2$ centered at $\boldsymbol{x}_1 = (-6, 0)$, and two smaller clusters with $\sigma_2 = \sigma_3 = 0.5$ centered at $\boldsymbol{x}_2 = (0, 0)$ and $\boldsymbol{x}_3 = (2, 0)$. We present the results of both the NJW algorithm [6] and the ZP algorithm [10] for two different weight vectors.

**Example I:** Weights $(\alpha_1, \alpha_2, \alpha_3) = (1/3, 1/3, 1/3)$. In the top left panel of fig. 2, $n = 1000$ random points from this density clearly show the difference in scales between the large cluster and the smaller ones. The first few eigenvectors of $M$ with a uniform $\sigma = 1$ are shown in the first two rows of the figure. The second eigenvector $\psi_2$ is indeed approximately piecewise constant and easily separates the larger cluster from the smaller ones. However, $\psi_3$ and $\psi_4$ are constant on the smaller clusters, capturing the relaxation process in the larger cluster ($\psi_3$ captures relaxation along the $y$-direction, hence it is not a function of the $x$-coordinate). In this example, only $\psi_5$ can separate the two small clusters. Therefore, as predicted theoretically, the NJW algorithm [6] fails to produce reasonable clusterings for all values of $\sigma$. In this example, the density of the large cluster is *low*, and therefore as expected and shown in the last row of fig. 2, the ZP algorithm clusters correctly.

**Example II:** Weights $(\alpha_1, \alpha_2, \alpha_3) = (0.8, 0.1, 0.1)$. In this case the density of the large cluster is high, and comparable to that of the small clusters. Indeed, as seen in fig. 3 and predicted theoretically

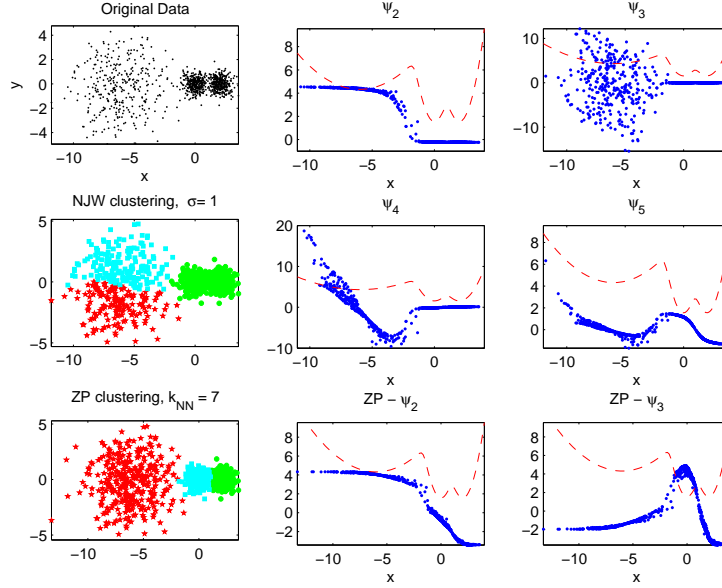

Figure 2: A three cluster dataset corresponding to example I (top left), clustering results of NJW and ZP algorithms [6, 10] (center and bottom left, respectively), and various eigenvectors of $M$ vs. the $x$ coordinate (blue dots in 2nd and 3rd columns). The red dotted line is the potential $U(x, 0)$.

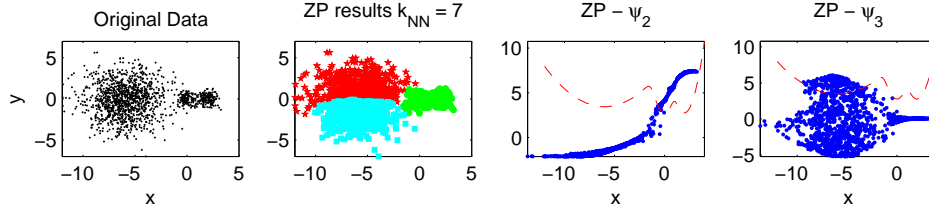

Figure 3: Dataset corresponding to example II and result of ZP algorithm.

the ZP algorithm fails to correctly cluster this data for all values of the parameter $k_{NN}$ in their algorithm. Needless to say, the NJW algorithm also fails to correctly cluster this example.

**Example III:** Consider data $\{x_i\}$ uniformly sampled from a domain $\Omega \subset \mathbb{R}^2$, which consists of three clusters, one a large rectangular container and two smaller disks, all connected by long and narrow tubes (see fig. 4 (left)). In this example the container is so large that the relaxation time inside it is slower than the characteristic time to diffuse between the small disks, hence NJW algorithm fails to cluster correctly. Since density is uniform, the ZP algorithm fails as well, fig. 4 (right).

Note that spectral clustering with the eigenvectors of the standard graph Laplacian has similar limitations, since the Euclidean distance between these eigenvectors is equal to the mean commute time on the graph [11]. Therefore, these methods may also fail when confronted with multiscale data.

## 5 Clustering with a Relaxation Time Coherence Measure

The analysis and examples of Sections 3 and 4 may suggest the use of more than $k$ eigenvectors in spectral clustering. However, clustering with $k$-means using 5 eigenvectors on the examples of Section 4 produced unsatisfactory results (not shown). Moreover, since the eigenvectors of the matrix $M$ are orthonormal under a specific weight function, they become increasingly oscillatory. Therefore, it is quite difficult to use them to detect a small cluster, much in analogy to Fourier analysis, where it is difficult to detect a localized bump in a function from its Fourier coefficients.

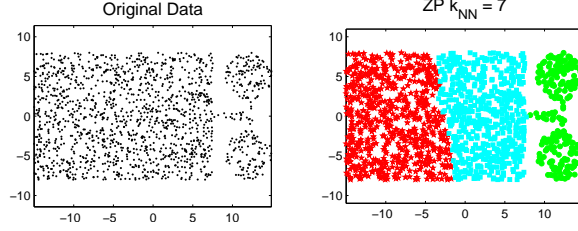

Figure 4: Three clusters defined solely by geometry, and result of ZP clustering (Example III).

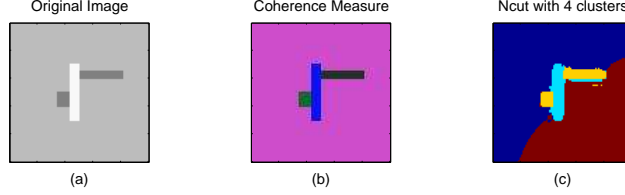

Figure 5: Normalized cut and coherence measure segmentation on a synthetic image.

Based on our analysis, we propose a different approach to graph-based clustering. Given the importance of relaxation times on the graph as indication of clusters, we propose a novel and principled measure for the coherence of a set of points as belonging to a single cluster. Our coherence measure can be used in conjunction with any clustering algorithm. Specifically, let $G = (V, W)$ be a weighted graph of points and let $V = S \cup (V \setminus S)$ be a possible partition (computed by some clustering algorithm). Our aim is to construct a meaningful measure to decide whether to accept or reject this partition. To this end, let $\lambda_2$ denote the second largest eigenvalue of the Markov matrix $M$ corresponding to the full graph $G$. We define $\tau_V = 1/(1 - \lambda_2)$ as the characteristic relaxation time of this graph. Similarly, $\tau_1$ and $\tau_2$ denote the characteristic relaxation times of the two subgraphs corresponding to the partitions $S$ and $V \setminus S$. If $V$ is a single coherent cluster, then we expect $\tau_V = O(\tau_1 + \tau_2)$. If, however, $V$ consists of two weakly connected clusters defined by $S$ and $V \setminus S$, then $\tau_1$ and $\tau_2$ measure the characteristic relaxation times inside these two clusters while $\tau_V$ measures the overall relaxation time. If the two sub-clusters are of comparable size, then $\tau_V \gg (\tau_1 + \tau_2)$. If however, one of them is much smaller than the other, then we expect $\max(\tau_1, \tau_2)/\min(\tau_1, \tau_2) \gg 1$. Thus, we define a set $V$ as coherent if either $\tau_V < c_1(\tau_1 + \tau_2)$ or if $\max(\tau_1, \tau_2)/\min(\tau_1, \tau_2) < c_2$. In this case, $V$ is not partitioned further. Otherwise, the subgraphs $S$ and $V \setminus S$ need to be further partitioned and similarly checked for their coherence. While a theoretical analysis is beyond the scope of this paper, reasonable numbers that worked in practice are $c_1 = 1.8$ and $c_2 = 10$. We note that other works have also considered relaxation times for clustering with different approaches [21, 22].

We now present use of this coherence measure with normalized cut clustering on the third example of Section 4. The first partition of normalized cut on this data with $\sigma = 1$ separates between the large container and the two smaller disks. The relaxation times of the full graph and the two subgraphs are $(\tau_V, \tau_1, \tau_2) = (1350, 294, 360)$. These numbers indicate that the full dataset is *not* coherent, and indeed should be partitioned. Next, we try to partition the large container. Normalized cuts partitions the container roughly into two parts with $(\tau_V, \tau_1, \tau_2) = (294, 130, 135)$, which according to our coherence measure means that the big container is a single structure that should not be split. Finally, normalized cut on the two small disks correctly separates them giving $(\tau_V, \tau_1, \tau_2) = (360, 18, 28)$, which indicates that indeed the two disks should be split. Further analysis of each of the single disks by our measure shows that each is a coherent cluster. Thus, combination of our coherence measure with normalized cut not only clusters correctly, but also automatically finds the correct number of clusters, regardless of cluster scale. Similar results are obtained for the other examples in this paper.

Finally, our analysis also applies to image segmentation. In fig. 5(a) a synthetic image is shown. The segmentation results of normalized cuts [24] and of the coherence measure combined with [23] appear in panels (b) and (c). Results on a real image are shown in fig. 6. Each segments is

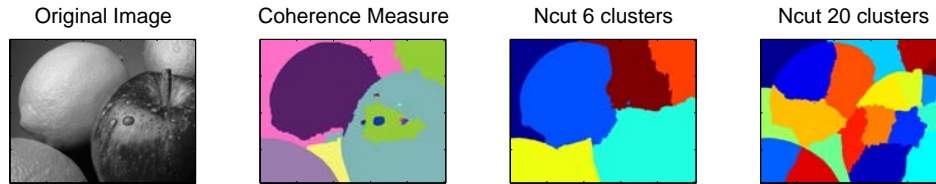

Figure 6: Normalized cut and coherence measure segmentation on a real image.

represented by a different color. With a small number of clusters normalized cut cannot find the small coherent segments in the image, whereas with a large number of clusters, large objects are segmented. Implementing our coherence measure with [23] finds salient clusters at different scales.

**Acknowlegments:** The research of BN was supported by the Israel Science Foundation (grant 432/06), by the Hana and Julius Rosen fund and by the William Z. and Eda Bess Novick Young Scientist fund.

## Footnotes

*Corresponding author. www.wisdom.weizmann.ac.il/~nadler

# References

[1] J. Shi and J. Malik. Normalized cuts and image segmentation, *PAMI*, Vol. 22, 2000.

[2] R. Kannan, S. Vempala, A. Vetta, On clusterings: good, bad and spectral, *J. ACM*, 51(3):497-515, 2004.

[3] D. Cheng, R. Kannan, S. Vempala, G. Wang, A divide and merge methodology for clustering, ACM SIGMOD/PODS, 2005.

[4] F.R.K. Chung, *Spectral Graph Theory*, Regional Conference Series in Mathematics Vol. 92, 1997.

[5] Y. Weiss, Segmentation using eigenvectors: a unifying view, *ICCV 1999*.

[6] A.Y. Ng, M.I. Jordan, Y. Weiss, On Spectral Clustering: Analysis and an algorithm, NIPS Vol. 14, 2002.

[7] N. Cristianini, J. Shawe-Taylor, J. Kandola, Spectral kernel methods for clustering, NIPS, Vol. 14, 2002.

[8] M. Belkin and P. Niyogi. Laplacian eigenmaps and spectral techniques for embedding and clustering, NIPS Vol. 14, 2002.

[9] S. Yu and J. Shi. Multiclass spectral clustering. ICCV 2003.

[10] L. Zelnik-Manor, P. Perona, Self-Tuning spectral clustering, NIPS, 2004.

[11] M. Saerens, F. Fouss, L. Yen and P. Dupont, The principal component analysis of a graph and its relationships to spectral clustering. ECML 2004.

[12] M. Meila, J. Shi. A random walks view of spectral segmentation, *AI and Statistics*, 2001.

[13] B. Nadler, S. Lafon, R.R. Coifman, I.G. Kevrekidis, Diffusion maps spectral clustering and eigenfunctions of Fokker-Planck operators, NIPS, 2005.

[14] S. Lafon, A.B. Lee, Diffusion maps and coarse graining: a unified framework for dimensionality reduction, graph partitioning, and data set parameterization, *PAMI*, 28(9):1393-1403, 2006.

[15] D. Harel and Y. Koren, On Clustering Using Random Walks, *FST TCS*, 2001.

[16] I. Fischer, J. Poland, Amplifying the block matrix structure for spectral clustering, Proceedings of the 14th Annual Machine Learning Conference of Belgium and the Netherlands, pp. 21-28, 2005.

[17] J. Malik, S. Belongie, T. Leung, J. Shi, Contour and texture analysis for image segmentation, *Int. J. Comp. Vis.* 43(1):7-27, 2001.

[18] E. Sharon, A. Brandt, R. Basri, Segmentation and Boundary Detection Using Multiscale Intensity Measurements, CVPR, 2001.

[19] M. Galun, E. Sharon, R. Basri and A. Brandt, Texture Segmentation by Multiscale Aggregation of Filter Responses and Shape Elements, *ICCV*, 2003.

[20] C.W. Gardiner, *Handbook of stochastic methods*, third edition, Springer NY, 2004.

[21] N. Tishby, N. Slonim, Data clustering by Markovian relaxation and the information bottleneck method, *NIPS*, 2000.

[22] C. Chennubhotla, A.J. Jepson, Half-lives of eigenflows for spectral clustering, *NIPS*, 2002.

[23] E. Sharon, A. Brandt, R. Basri, Fast multiscale image segmentation, *ICCV*, 2000.

[24] T. Cour, F. Benezit, J. Shi. Spectral Segmentation with Multiscale Graph Decomposition. *CVPR*, 2005.